# Gradients for retinotectal mapping

**Geoffrey J. Goodhill**
Georgetown Institute for Cognitive and Computational Sciences
Georgetown University Medical Center
3970 Reservoir Road
Washington DC 20007
geoff@giccs.georgetown.edu

## Abstract

The initial activity-independent formation of a topographic map in the retinotectal system has long been thought to rely on the matching of molecular cues expressed in gradients in the retina and the tectum. However, direct experimental evidence for the existence of such gradients has only emerged since 1995. The new data has provoked the discussion of a new set of models in the experimental literature. Here, the capabilities of these models are analyzed, and the gradient shapes they predict in vivo are derived.

## 1 Introduction

During the early development of the visual system in for instance rats, fish and chickens, retinal axons grow across the surface of the optic tectum and establish connections so as to form an ordered map. Although later neural activity refines the map, it is not required to set up the initial topography (for reviews see Udin & Fawcett (1988); Goodhill (1992)). A long-standing idea is that the initial topography is formed by matching gradients of receptor expression in the retina with gradients of ligand expression in the tectum (Sperry, 1963). Particular versions of this idea have been formalized in theoretical models such as those of Prestige & Willshaw (1975), Willshaw & von der Malsburg (1979), Whitelaw & Cowan (1981), and Gierer (1983;1987). However, these models were developed in the absence of any direct experimental evidence for the existence of the necessary gradients. Since 1995, major breakthroughs have occurred in this regard in the experimental literature. These center around the Eph (Erythropoetin-producing hepatocellular) subfamily of receptor tyrosine kinases. Eph receptors and their ligands have been shown to be expressed in gradients in the developing retina and tectum respectively, and to play a role in guiding axons to appropriate positions. These exciting new developments have led experimentalists to discuss theoretical models differ-

ent from those previously proposed (e.g. Tessier-Lavigne (1995); Tessier-Lavigne & Goodman (1996); Nakamoto et al, (1996)). However, the mathematical consequences of these new models, for instance the precise gradient shapes they require, have not been analyzed. In this paper, it is shown that only certain combinations of gradients produce appropriate maps in these models, and that the validity of these models is therefore experimentally testable.

## 2 Recent experimental data

Receptor tyrosine kinases are a diverse class of membrane-spanning proteins. The Eph subfamily is the largest, with over a dozen members. Since 1990, many of the genes encoding Eph receptors and their ligands have been shown to be expressed in the developing brain (reviewed in Friedman & O'Leary, 1996). Ephrins, the ligands for Eph receptors, are all membrane anchored. This is unlike the majority of receptor tyrosine kinase ligands, which are usually soluble. The ephrins can be separated into two distinct groups A and B, based on the type of membrane anchor. These two groups bind to distinct sets of Eph receptors, which are thus also called A and B, though receptor-ligand interaction is promiscuous within each subgroup. Since many research groups discovered members of the Eph family independently, each member originally had several names. However a new standardized notation was recently introduced (Eph Nomenclature Committee, 1997), which is used in this paper.

With regard to the mapping from the nasal-temporal axis of the retina to the anterior-posterior axis of the tectum (figure 1), recent studies have shown the following (see Friedman & O'Leary (1996) and Tessier-Lavigne & Goodman (1996) for reviews).

- EphA3 is expressed in an increasing nasal to temporal gradient in the retina (Cheng et al, 1995).

- EphA4 is expressed uniformly in the retina (Holash & Pasquale, 1995).

- Ephrin-A2, a ligand of both EphA3 and EphA4, is expressed in an increasing rostral to caudal gradient in the tectum (Cheng et al, 1995).

- Ephrin-A5, another ligand of EphA3 and EphA4, is also expressed in an increasing rostral to caudal gradient in the tectum, but at very low levels in the rostral half of the tectum (Drescher et al, 1995).

All of these interactions are *repulsive*. With regard to mapping along the complementary dimensions, EphB2 is expressed in a high ventral to low dorsal gradient in the retina, while its ligand ephrin-B1 is expressed in a high dorsal to low ventral gradient in the tectum (Braisted et al, 1997). Members of the Eph family are also beginning to be implicated in the formation of topographic projections between many other pairs of structures in the brain (Renping Zhou, personal communication). For instance, EphA5 has been found in an increasing lateral to medial gradient in the hippocampus, and ephrin-A2 in an increasing dorsal to ventral gradient in the septum, consistent with a role in establishing the topography of the map between hippocampus and septum (Gao et al, 1996).

The current paper focusses just on the paradigm case of the nasal-temporal to anterior-posterior axis of the retinotectal mapping. Actual gradient shapes in this system have not yet been quantified. The analysis below will assume that certain gradients are linear, and derive the consequences for the other gradients.

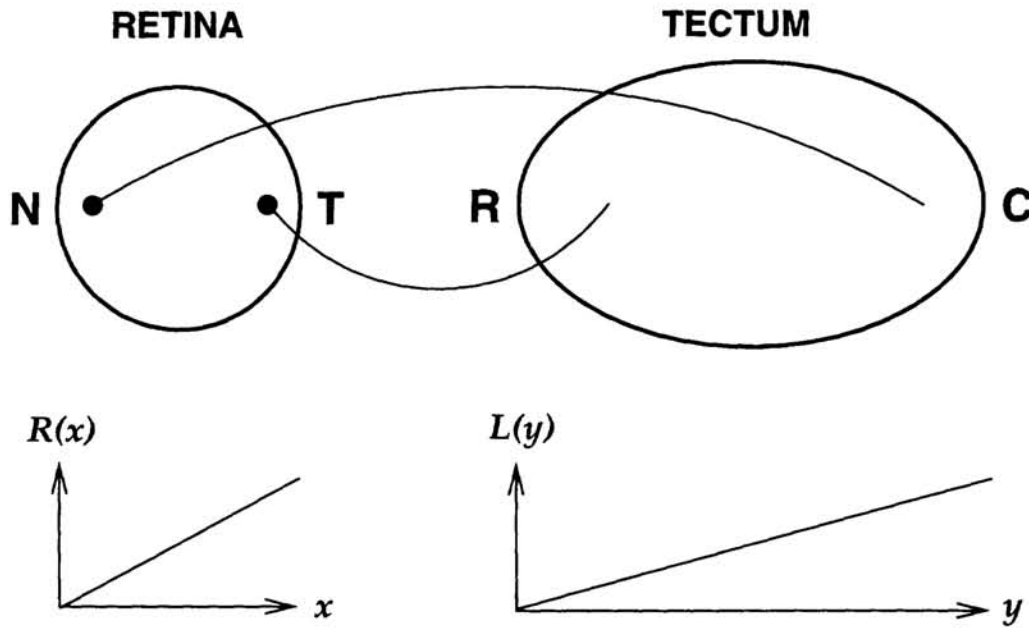

Figure 1: This shows the mapping that is normally set up from the retina to the tectum. Distance along the nasal-temporal axis of the retina is referred to as $x$ and receptor concentration as $R(x)$. Distance along the rostral-caudal axis of the tectum is referred to as $y$ and ligand concentration as $L(y)$.

## 3   Mathematical models

Let $R$ be the concentration of a receptor expressed on a growth cone or axon, and $L$ the concentration of a ligand present in the tectum. Refer to position along the nasal-temporal axis of the retina as $x$, and position along the rostral-caudal axis of the tectum as $y$, so that $R = R(x)$ and $L = L(y)$ (see figure 1). Gierer (1983; 1987) discusses how topographic information could be signaled by interactions between ligands and receptors. A particular type of interaction, proposed by Nakamoto et al (1996), is that the concentration of a "topographic signal", the signal that tells axons where to stop, is related to the concentration of receptor and ligand by the law of mass action:

$$G(x,y) = kR(x)L(y) \tag{1}$$

where $G(x,y)$ is the concentration of topographic signal produced within an axon originating from position $x$ in the retina when it is at position $y$ in the tectum, and $k$ is a constant. In the general case of multiple receptors and ligands, with promiscuous interactions between them, this equation becomes

$$G(x,y) = \sum_{i,j} k_{ij} R_i(x) L_j(y) \tag{2}$$

Whether each receptor-ligand interaction is attractive or repulsive is taken care of by the sign of the relevant $k_{ij}$.

Two possibilities for how $G(x,y)$ might produce a stop (or branch) signal in the growth cone (or axon) are that this occurs when (1) a "set point" is reached (discussed in, for example, Tessier-Lavigne & Goodman (1996); Nakamoto et al (1996)), i.e. $G(x,y) = c$ where $c$ is a constant, or (2) attraction (or repulsion) reaches a local maximum (or minimum), i.e. $\frac{\partial G(x,y)}{\partial y} = 0$ (Gierer, 1983; 1987). For a smooth, uni-

form mapping, one of these conditions must hold along a line $y \propto x$. For simplicity assume the constant of proportionality is unity.

## 3.1  Set point rule

For one gradient in the retina and one gradient in the tectum (i.e. equation 1), this requires that the ligand gradient be inversely proportional to the receptor gradient:

$$L(x) = \frac{c}{R(x)}$$

If $R(x)$ is linear (c.f. the gradient of EphA3 in the retina), the ligand concentration is required to go to infinity at one end of the tectum (see figure 2). One way round this is to assume $R(x)$ does not go to zero at $x = 0$: the experimental data is not precise enough to decide on this point. However, the addition of a second receptor gradient gives

$$L(x) = \frac{c}{k_1 R_1(x) + k_2 R_2(x)}$$

If $R_1(x)$ is linear and $R_2(x)$ is flat (c.f. the gradient of EphA4 in the retina), then $L(y)$ is no longer required to go to infinity (see figure 2). For two receptor and two ligand gradients many combinations of gradient shapes are possible. As a special case, consider $R_1(x)$ linear, $R_2(x)$ flat, and $L_1(y)$ linear (c.f. the gradient of Elf1 in the tectum). Then $L_2$ is required to have the shape

$$L_2(y) = \frac{ay^2 + by}{dy + e}$$

where $a$, $b$, $d$, $e$ are constants. This shape depends on the values of the constants, which depend on the relative strengths of binding between the different receptor and ligand combinations. An interesting case is where $R_1$ binds only to $L_1$ and $R_2$ binds only to $L_2$, i.e. there is no promiscuity. In this case we have

$$L_2(y) \propto y^2$$

(see figure 2). This function somewhat resembles the shape of the gradient that has been reported for ephrin-A5 in the tectum. However, this model requires one gradient to be attractive, whereas both are repulsive.

## 3.2  Local optimum rule

For one retinal and one tectal gradient we have the requirement

$$R(x)\frac{\partial L(y)}{\partial y} = 0$$

This is not generally true along the line $y = x$, therefore there is no map. The same problem arises with two receptor gradients, whatever their shapes. For two receptor and two ligand gradients many combinations of gradient shapes are possible. (Gierer (1983; 1987) investigated this case, but for a more complicated reaction law for generating the topographic signal than mass action.) For the special case introduced above, $L_2(y)$ is required to have the shape

$$L_2(y) = ay + b\log(dy + e) + f$$

where $a$, $b$, $d$, $e$, and $f$ are constants as before. Considering the case of no promiscuity, we again obtain

$$L_2(y) \propto y^2$$

i.e. the same shape for $L_2(y)$ as that specified by the set point rule.

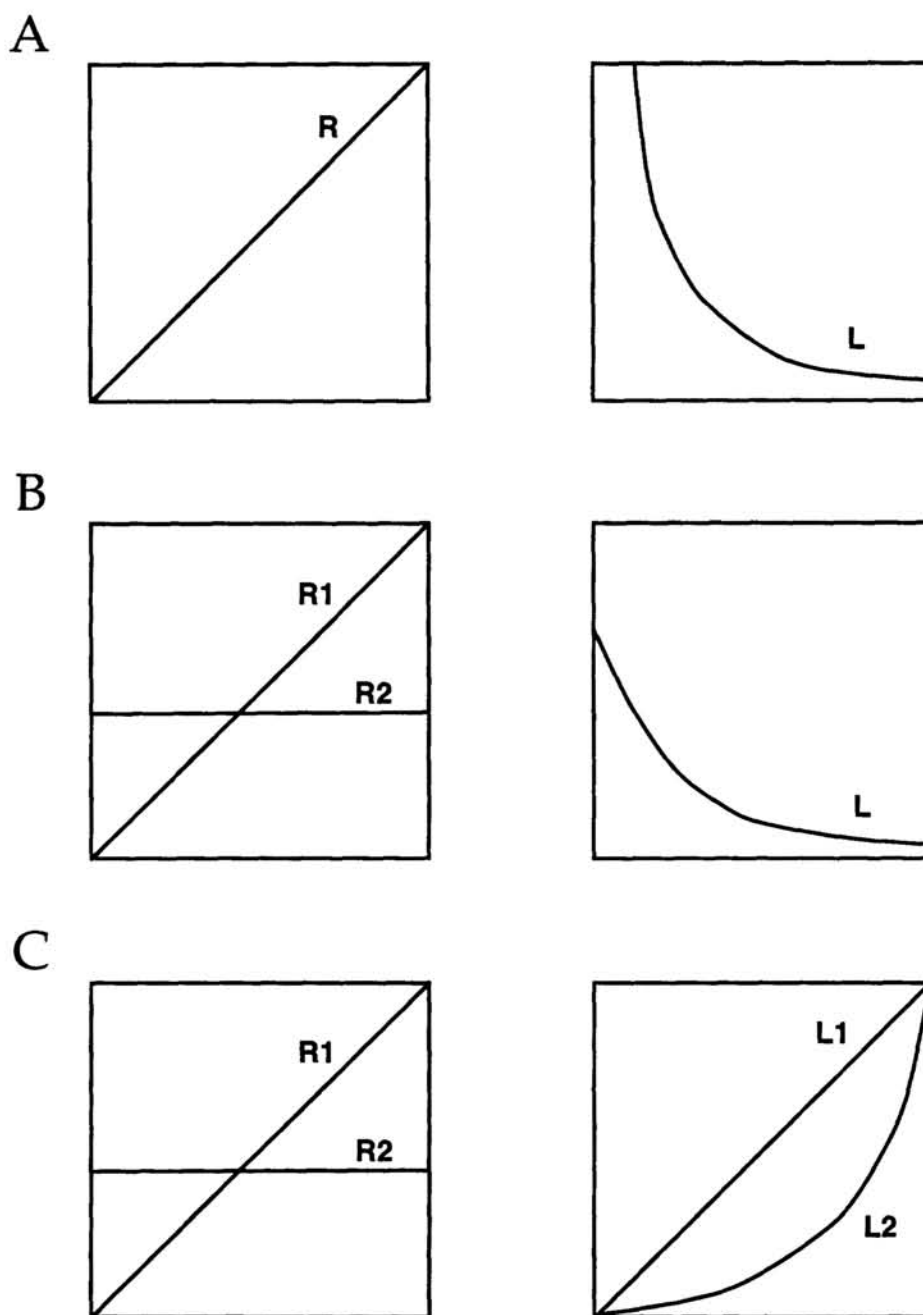

Figure 2: Three combinations of gradient shapes that are sufficient to produce a smooth mapping with the mass action rule. In the left column the horizontal axis is position in the retina while the vertical axis is the concentration of receptor. In the right column the horizontal axis is position in the tectum while the vertical axis is the concentration of ligand. Models A and B work with the set point but not the local optimum rule, while model C works with both rules. For models B and C, one gradient is negative and the other positive.

# 4 Discussion

For both rules, there is a set of gradient shapes for the mass-action model that is consistent with the experimental data, except for the fact that they require one gradient in the tectum to be attractive. Both ephrin-A2 and ephrin-A5 have repulsive effects on their receptors expressed in the retina, which is clearly a problem for these models. The local optimum rule is more restrictive than the set point rule, since it requires at least two ligand gradients in the tectum. However, unlike the set point rule, it supplies directional information (in terms of an appropriate gradient for the topographic signal) when the axon is not at the optimal location.

In conclusion, models based on the mass action assumption in conjunction with either a "set point" or "local optimum" rule can be true only if the relevant gradients satisfy the quantitative relationships described above. A different theoretical approach, which analyzes gradients in terms of their ability to guide axons over the maximum possible distance, also makes predictions about gradient shapes in the retinotectal system (Goodhill & Baier, 1998). Advances in experimental technique should enable a more quantitative analysis of the gradients in situ to be performed shortly, allowing these predictions to be tested. In addition, analysis of particular Eph and ephrin knockout mice (for instance ephrin-A5 (Yates et al, 1997)) is now being performed, which should shed light on the role of these gradients in normal map development.

# Bibliography

Braisted, J.E., McLaughlin, T., Wang, H.U., Friedman, G.C., Anderson, D.J. & O'Leary, D.D.M. (1997). Graded and lamina-specific distributions of ligands of EphB receptor tyrosine kinases in the developing retinotectal system. *Developmental Biology*, **191** 14-28.

Cheng, H.J., Nakamoto, M., Bergemann, A.D & Flanagan, J.G. (1995). Complementary gradients in expression and binding of Elf-1 and Mek4 in development of the topographic retinotectal projection map. *Cell*, **82**, 371-381.

Drescher, U., Kremoser, C., Handwerker, C., Loschinger, J., Noda, M. & Bonhoeffer, F. (1995). In-vitro guidance of retinal ganglion-cell axons by RAGS, a 25 KDa tectal protein related to ligands for Eph receptor tyrosine kinases. *Cell*, **82**, 359-370.

Eph Nomenclature Committee (1997). Unified nomenclature for Eph family receptors and their ligands, the ephrins. *Cell*, **90**, 403-404.

Friedman, G.C. & O'Leary, D.D.M. (1996). Eph receptor tyrosine kinases and their ligands in neural development. *Curr. Opin. Neurobiol.*, **6**, 127-133.

Gierer, A. (1983). Model for the retinotectal projection. *Proc. Roy. Soc. Lond. B*, **218**, 77-93.

Gierer, A. (1987). Directional cues for growing axons forming the retinotectal projection. *Development*, **101**, 479-489.

Gao, P.-P., Zhang, J.-H., Yokoyama, M., Racey, B., Dreyfus, C.F., Black, I.B. & Zhou, R. (1996). Regulation of topographic projection in the brain: Elf-1 in the hippocampalseptal system. *Proc. Nat. Acad. Sci. USA*, **93**, 11161-11166.

Goodhill, G.J. (1992). *Correlations, Competition and Optimality: Modelling the Development of Topography and Ocular Dominance*. Cognitive Science Research Paper CSRP 226, University of Sussex. Available from www.giccs.georgetown.edu/~geoff

Goodhill, G.J. & Baier, H. (1998). Axon guidance: stretching gradients to the limit. *Neural Computation,* in press.

Holash, J.A. & Pasquale, E.B. (1995). Polarized expression of the receptor protein-tyrosine kinase Cek5 in the developing avian visual system. *Developmental Biology,* **172,** 683-693.

Nakamoto, M., Cheng H.J., Friedman, G.C., Mclaughlin, T., Hansen, M.J., Yoon, C.H., O'Leary, D.D.M. & Flanagan, J.G. (1996). Topographically specific effects of ELF-1 on retinal axon guidance in-vitro and retinal axon mapping in-vivo. *Cell,* **86,** 755-766.

Prestige, M.C. & Willshaw, D.J. (1975). On a role for competition in the formation of patterned neural connexions. *Proc. R. Soc. Lond. B,* **190,** 77-98.

Sperry, R.W. (1963). Chemoaffinity in the orderly growth of nerve fiber patterns and connections. *Proc. Nat. Acad. Sci., U.S.A.,* **50,** 703-710.

Tessier-Lavigne, M. (1995). Eph receptor tyrosine kinases, axon repulsion, and the development of topographic maps. *Cell,* **82,** 345-348.

Tessier-Lavigne, M. and Goodman, C.S. (1996). The molecular biology of axon guidance. *Science,* **274,** 1123-1133.

Udin, S.B. & Fawcett, J.W. (1988). Formation of topographic maps. *Ann. Rev. Neurosci.,* **11,** 289-327.

Whitelaw, V.A. & Cowan, J.D. (1981). Specificity and plasticity of retinotectal connections: a computational model. *Jou. Neurosci.,* **1,** 1369-1387.

Willshaw, D.J. & Malsburg, C. von der (1979). A marker induction mechanism for the establishment of ordered neural mappings: its application to the retinotectal problem. *Phil. Trans. Roy. Soc. B,* **287,** 203-243.

Yates, P.A., McLaughlin, T., Friedman, G.C., Frisen, J., Barbacid, M. & O'Leary, D.D.M. (1997). Retinal axon guidance defects in mice lacking ephrin-A5 (AL-1/RAGS). *Soc. Neurosci. Abstracts,* **23,** 324.